# Bayes Networks on Ice: Robotic Search for Antarctic Meteorites

**Liam Pedersen**[*], **Dimi Apostolopoulos, Red Whittaker**
Robotics Institute
Carnegie Mellon University
Pittsburgh, PA 15213
*{pedersen+, da1v, red}@ri.cmu.edu*

## Abstract

A Bayes network based classifier for distinguishing terrestrial rocks from meteorites is implemented onboard the Nomad robot. Equipped with a camera, spectrometer and eddy current sensor, this robot searched the ice sheets of Antarctica and autonomously made the first robotic identification of a meteorite, in January 2000 at the Elephant Moraine. This paper discusses rock classification from a robotic platform, and describes the system onboard Nomad.

## 1  Introduction

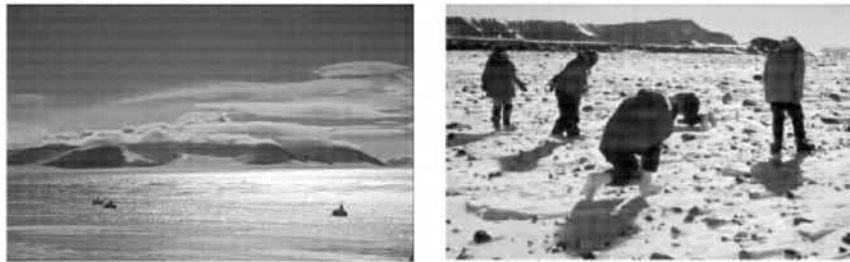

**Figure 1 :** Human meteorite search with snowmobiles on the Antarctic ice sheets, and on foot in the moraines.

Antarctica contains the most fertile meteorite hunting grounds on Earth. The pristine, dry and cold environment ensures that meteorites deposited there are preserved for long periods. Subsequent glacial flow of the ice sheets where they land concentrates them in particular areas. To date, most meteorites recovered throughout history have been done so in Antarctica in the last 20 years. Furthermore, they are less likely to be contaminated by terrestrial compounds.

---

[*] http://www.cs.cmu.edu/~pedersen

Meteorites are of interest to space scientists because, with the exception of the Apollo lunar samples, they are the sole source of extra-terrestrial material and a window on the early evolution of the solar system. The identification of Martian and lunar meteorite samples, and the (controversial) evidence of fossil bacteria in the former underscores the importance of systematically retrieving as many samples as possible.

Currently, Antarctic meteorite samples are collected by human searchers, either on foot, or on snowmobiles, who systematically search an area and retrieve samples according to strict protocols. In certain blue ice fields the only rocks visible are meteorites. At other places (moraines – areas where the ice flow brings rocks to the surface) searchers have to contend with many terrestrial rocks (Figure 1).

## 1.1 Robotic search for Antarctic meteorites

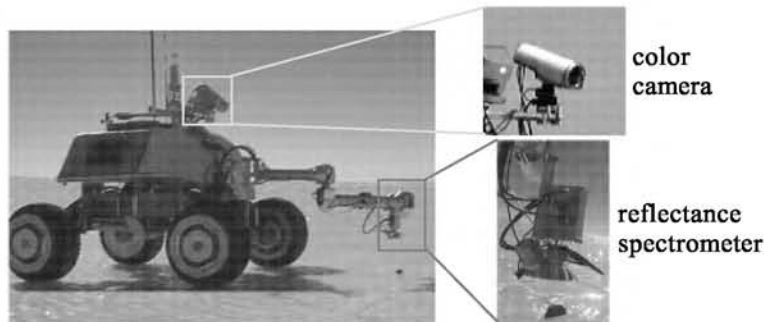

**Figure 2 :** Nomad robot, equipped with scientific instruments, investigates a rock in Antarctica.

With the goal of autonomously search for meteorites in Antarctica, Carnegie Mellon University has built and demonstrated [1] a robot, Nomad (Figure 2), capable of long duration missions in harsh environments. Nomad is equipped with a color camera on a pan-tilt platform to survey the ice for rocks and acquire close up images of any candidate objects, and a manipulator arm to place the fiber optic probe of a specially designed visible light reflectance spectrometer over a sample. The manipulator arm can also place other sensors, such a metal detector.

The eventual goal, beyond Antarctic meteorite search, is to develop technologies for extended robotic exploration of remote areas, including planetary surfaces. One particular technology is the capacity to carry out autonomous science, including autonomous geology and the ability to recognize a broad range of rock types and note exceptions.

Identifying meteorites amongst terrestrial rocks is the fundamental engineering problem of robotic meteorite search and is the topic addressed by the rest of this paper.

## 2 Bayes network rock and meteorite classifier

Classifying rocks from a mobile robotic vehicle entails several unique issues:

- The classifier must learn from examples. Human experts often have trouble explaining how they can identify many rocks, and will refer to an example. In the words of a veteran Antarctic meteorite searcher [2] "First you find a few meteorites, then you know what to look for".

A complication is the difficulty of acquiring large sets of training data, *under realistic field conditions*. To date this has required two earlier expeditions to Antarctica, as well as visits to the Arctic and the Atacama desert in Chile. Therefore, it is necessary to constrain a classifier as much as possible with available prior knowledge, so that training can be accomplished with minimum data.

- The classifier must be able to accept incomplete data, and compound evidence for different hypotheses as more information becomes available. The robot has multiple sensors, and there is a cost associated with using each one. Sensors such as the spectrometer are particularly expensive to use because the robot must be maneuvered to bring the rock sample into the sensor manipulator workspace. Therefore, it is desirable that initial classifications be made using data from cheap long range sensors, such as a color camera, before final verification using expensive sensors on promising rock samples.

  A corollary of this is that the classifier should accept prior evidence from other sources, such as an experts knowledge on what to expect in a particular location.

- Rock classes are often ambiguous, and the distinctions between certain types fuzzy at best [3]. The classifier must handle this ambiguity, and indicate several likely hypotheses if a definite classification cannot be achieved.

These requirements for a robotic rock classifier argue strongly in favor of a Bayes network based approach, which can satisfy them all. The intuitive graphical structure of a Bayes network makes it easier to encode physical constraints into the network topology, thus reducing the intrinsic dimensionality. Bayesian update is a principled way to compound evidence, and prior information is naturally represented by prior probabilities.

Additionally, with a Bayes network it is simple to compute the likelihood of any new data, and thus conceivably recognize bad sensor readings. Furthermore, the network can be queried to estimate the information gain of further sensor readings, enabling active sensor selection.

## 2.1 Network architecture

The (simplified) network architecture for distinguishing rocks from meteorites, using features from sensor data, is shown in Figure 3. It is a compromise between a fully connected network (no constraints whatsoever, and computationally intractable) and a naïve Bayes classifier (can be efficiently evaluated, but lacks sufficient representational power). Sensor features are only weakly (conditionally) dependent on each other because of a careful choice of suitable features, and the intermediate node **Rock-type**, whose states include all possible rock and meteorite types likely to be encountered by the classifier.

A complication is that the sensor features are continuous quantities, yet the Bayes network implementation can only handle discrete variables. Therefore the continuous variables need to be suitably quantized.

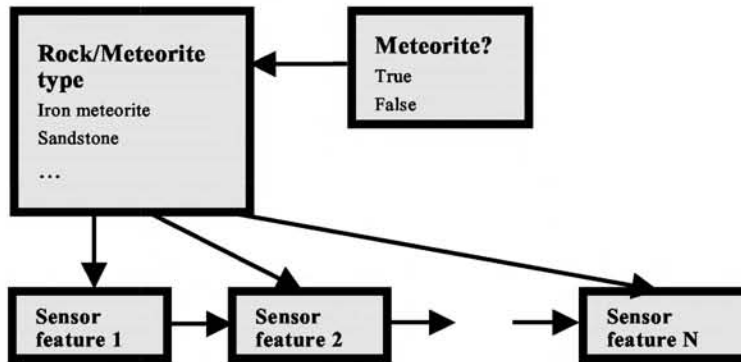

**Figure 3 :** Bayes network for discriminating meteorites and rocks based on features computed from sensor data.

## 2.2 Sensors and feature vectors

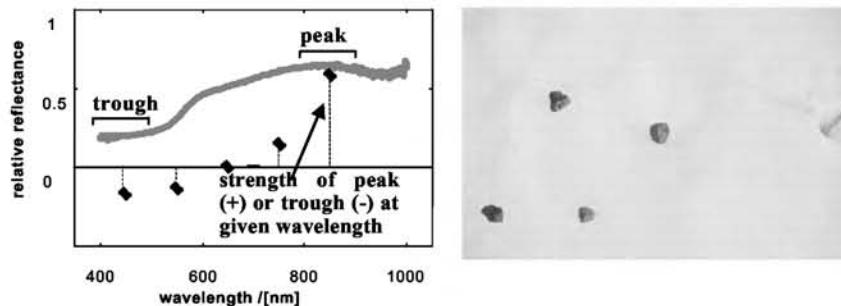

**Figure 4 :** Example spectrum (with extracted features) and color images of rocks on ice. One of the rocks in the image is meteorite.

In Antarctica Nomad acquired reflectance spectra and color images (Figure 4) of sample rocks. Spectra are obtained by shining white light on the sample and analyzing the reflected light to determine the fraction of light reflected at a series of wavelengths.

The relevant features in a spectrum, for the purpose of identifying rocks, are the presence, location and size of peaks and troughs in the spectrum (Figure 4), and the average magnitude (albedo) of the spectrum over certain wavelengths. Spectral troughs and peaks are detected by computing the correlation of the spectrum with a set of 10 templates over a finite region of support (50 nm). Restricting the degree of overlap between templates minimizes statistical dependencies between the resulting spectral features (Figure 3). Normalizing the correlation coefficients makes them (conditionally) independent of the average spectral intensity and robust to changes to scale (important, because in practice, when making a field measurement of a spectrum it is difficult to accurately determine the scale). A 13 element real valued feature vector (each component corresponding to a sensor feature node in Figure 3) is thus obtained from the original 1000+ element spectrum.

Color images are harder to interpret (one of the rocks in Figure 4 is a meteorite). First the rock needs to be segmented from the background of snow and ice in the

image, using a partially observable Markov model [4]. Features of interest are the rock cross sectional area (used as a proxy for size, and requiring that the scaling of the images be known), average color, and simple texture and shape metrics [4]. Meteorites tend to be small and dark compared to terrestrial rocks. An 8 element real valued feature vector is computed from each image.

All real valued features are quantized prior to being entered into the Bayes network, which cannot handle continuous quantities.

## 2.3 Network training

The conditional probability matrices (CPM's) describing the probability distributions of network sensor feature nodes given **Rock type** (and other parent nodes) are learned from examples (of rock types along with the associated feature vectors derived from sensor readings on rock samples of the given type) using the algorithm in [5]. If **X** is a node (with N states) with parent **Y**, and with CPM $P_{ij} = P(X=i|Y=j)$, then each column is represented by a Dirichlet distribution (initially uniform) and assumed independent of the others. If $\alpha_1..\alpha_N$ are the Dirichlet parameters for $P(X|Y=j)$ then $P_{ij} = \alpha_{ik}\Big/\sum_k \alpha_k$ [6]. Given a new example $\{X=i,Y=j\}$ with weight $w$ the Dirichlet parameters are updated: $\alpha_i \rightarrow \alpha_i + w$. This is a true Bayesian learning algorithm, and is stable. Furthermore, it is possible to weight each training sample to reflect its frequency of occurrence for the rock type that generated it. This is especially important if multiple sensor readings are taken from a single sample

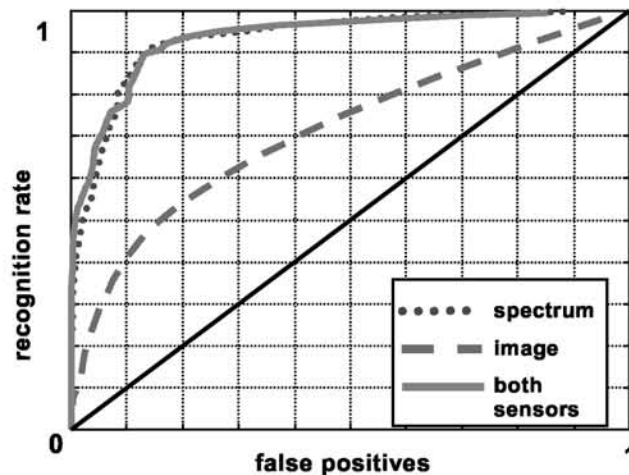

**Figure 5 :** Classifier rate of classification curves using laboratory data for training and testing (25% cross validation), for different sensors.

The training data (gathered from previous Antarctic expeditions, and from US laboratory collections[*] of meteorites and Antarctic rocks) is insufficient to fully populate the (quantized) space on which the CPM's are defined, unless the real valued feature nodes are very coarsely quantized. To avoid this, more spectral data was generated from each sample spectra by adding random noise (generated by a

---

[*] Johnson Space Center, Houston and Ohio State University, Columbus.

non-linear spectrometer noise model) to it. (This is analogous to the approach used by [7] for training neural networks).

Using meteorite and terrestrial rock data acquired in the lab, partitioned into 75% training, 25% testing cross validation sets, the Rate of Classification (ROC) curves in Figure 5 are generated. Note the superior classification with spectra versus classification with color images only. In fact, given a spectrum, a color image does not improve classification. However, because it is easier to acquire color images than spectra, they are still useful as a sensor for preliminary screening.

## 3   Antarctica 2000 field results

In January 2000 the Nomad robot was deployed to the Elephant moraine in Antarctica for robotic meteorite searching trials. Nomad searched areas known to contain meteorites, autonomously acquiring color images and reflection spectra of both native terrestrial rocks and meteorites, and classifying them. On January 22, 2000 Nomad successfully identified a meteorite amongst terrestrial rocks on the ice sheet (http://www.frc.ri.cmu.edu/projects/meteorobot2000/).

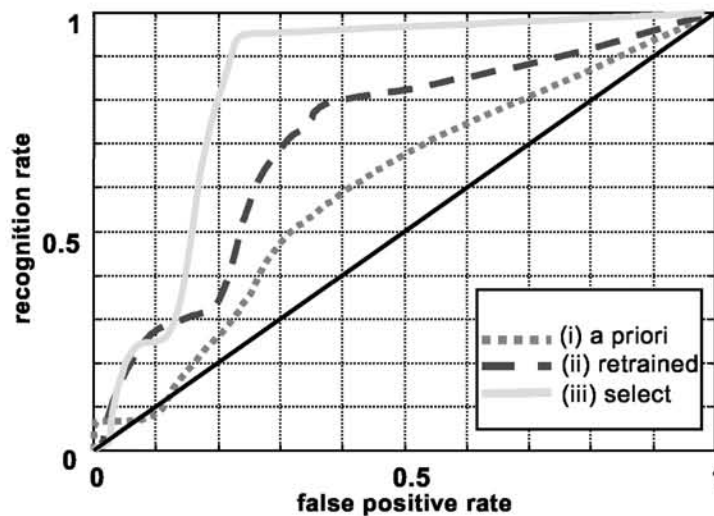

**Figure 6 :**  Rate of classification curves for the Nomad robot searching for meteorites in Antarctica, 2000 A.D.

Overall performance (using spectra only, due to a problem that developed with camera zoom control) is indicated by the ROC performance curves in Figure 6. These were generated from a test set of rocks and meteorites (40 and 4 samples respectively, with multiple readings of each) in a particular area of the moraine. Figure 6(i) is using the *a priori* classifier built from the lab data (used to generate Figure 5), acquired prior to arrival in Antarctica. Performance clearly does not match that in Figure 5. There is a notable improvement in (ii), the ROC curve for the same classifier further trained with field data acquired by the robot in the area (from 8 rocks and 2 meteorites *not* in this test set).

Even with retraining, classification is systematically bad for a particular class of rocks (hydro-thermally altered dolerites and basalts) that occurred in the Elephant moraine. These rocks are stained red with iron oxide (rust) whose spectrum has a very prominent peak at 900 nm, precisely where many meteorite spectra also have a peak. This is not surprising, given that most meteorites contain metallic iron, and

therefore can have rust on the surface. However, these rocks were absent from the initial training set and not initially expected in this area. Performance is much better if these rocks are removed from the test set (iii) and the retrained classifier is used.

# 4  Conclusions

With the caveat that training be continued using data acquired by the robot in the field, the Bayes network approach to robotic rock classification is a viable approach to this task. Nomad did autonomously identify several meteorites. However, in areas with hydro-thermally altered rocks (iron-oxide stained) the reflection spectrometer must be supplemented by other sensors, such as metal detectors, magnetometers or more exotic spectrometers (thermal emission or Raman), obviously at greater cost.

Sensor noise and systematic effects due to autonomous robot placement of sensors on samples in the unstructured and uncontrolled polar environment are significant. They are hard to know a priori and need to be learned from data acquired *by the robot*, and in field conditions, as demonstrated by the significant improvement in classification achieved after field retraining.

Further work needs to be done in selective sensor selection, active modeling of the local geographical distribution of rocks, and recognizing bad sensor readings, but indications are that this can be done in a principled way with the Bayes network classifier and will be addressed in future papers.

## Acknowledgments

The authors gratefully acknowledge the invaluable assistance of Professor William Cassidy of the University of Pittsburgh, Professor Gunter Faure of Ohio State University, Marilyn Lindstrom and the staff at the Antarctic meteorite curation facility of NASA's Johnson Space Center, and Drs. Martial Hebert and Andrew Moore of Carnegie Mellon University.

This work was funded by NASA, and supported in Antarctica by the National Science Foundation's Office of Polar Programs.

## References

[1] D. Apostolopoulos, M. Wagner, W. Whittaker, "Technology and Field Demonstration Results in the Robotic Search for Antarctic Meteorites", *Field and Service Robotics Conference*, Pittsburgh, USA, 1999

[2] Cassidy, William, University of Pittsburgh Department of Geology, personal communication, 1997.

[3] R. Dietrich and B. Skinner, *Rocks and Minerals*, Wiley 1979.

[4] L. Pedersen, D. Apostolopoulos, W. Whittaker, T. Roush, G. Benedix, "Sensing and Data Classification for Robotic Meteorite Search", Proceedings of *SPIE Photonics East Conference,* Boston, 1998.

[5] Spiegelhalter, David J., A. Philip Dawid, Steffen L. Lauritzen and Robert G. Cowell, "Bayesian analysis in expert systems" in *Statistical Science*, 8(3), p219-283., 1993.

[6] A. Gelman, J. Carlin, H. Stern, D. Rubin, *Bayesian Data Analysis*, Chapman & Hall, 1995.

[7] D. Pomerleau, "Efficient Training of Artificial Neural Networks for Autonomous Navigation", *NeurComp* vol. 3 no. 1 p 88-97, 1991
